# OPTIMIZATION WITH ARTIFICIAL NEURAL NETWORK SYSTEMS:
## A MAPPING PRINCIPLE
## AND
## A COMPARISON TO GRADIENT BASED METHODS [†]

Harrison MonFook Leong
Research Institute for Advanced Computer Science
NASA Ames Research Center 230-5
Moffett Field, CA, 94035

## ABSTRACT

General formulae for mapping optimization problems into systems of ordinary differential equations associated with artificial neural networks are presented. A comparison is made to optimization using gradient-search methods. The performance measure is the settling time from an initial state to a target state. A simple analytical example illustrates a situation where dynamical systems representing artificial neural network methods would settle faster than those representing gradient-search. Settling time was investigated for a more complicated optimization problem using computer simulations. The problem was a simplified version of a problem in medical imaging: determining loci of cerebral activity from electromagnetic measurements at the scalp. The simulations showed that gradient based systems typically settled 50 to 100 times faster than systems based on current neural network optimization methods.

## INTRODUCTION

Solving optimization problems with systems of equations based on neurobiological principles has recently received a great deal of attention. Much of this interest began when an artificial neural network was devised to find near-optimal solutions to an *np*-complete problem [13]. Since then, a number of problems have been mapped into the same artificial neural network and variations of it [10, 13, 14, 17, 18, 19, 21, 23, 24]. In this paper, a unifying principle underlying these mappings is derived for systems of first to $n^{th}$-order ordinary differential equations. This mapping principle bears similarity to the mathematical tools used to generate optimization methods based on the gradient. In view of this, it seemed important to compare the optimization efficiency of dynamical systems constructed by the neural network mapping principle with dynamical systems constructed from the gradient.

## THE PRINCIPLE

This paper concerns itself with networks of computational units having a state variable $v$, a function $f$ that describes how a unit is driven by inputs, a linear ordinary differential operator with constant coefficients $D(v)$ that describes the dynamical response of each unit, and a function $g$ that describes how the output of a computational unit is determined from its state $v$. In particular, the paper explores how outputs of the computational units evolve with time in terms of a scalar function $E$, a single state variable for the whole network. Fig. 1 summarizes the relationships between variables, functions, and operators associated with each computational unit. Eq. (1) summarizes the equations of motion for a network composed of such units:

$$\vec{D}^{(M)}(v) = \vec{f}(g_1(v_1), \ldots, g_N(v_N))  \tag{1}$$

where the $i^{th}$ element of $\vec{D}^{(M)}$ is $D^{(M)}(v_i)$, superscript $(M)$ denotes that operator $D$ is $M^{th}$ order, the $i^{th}$ element of $\vec{f}$ is $f_i(g_1(v_1), \ldots, g_N(v_N))$, and the network is comprised of $N$ computational units. The network of Hopfield [12] has $M=1$, functions $\vec{f}$ are weighted linear sums, and functions $\vec{g}$ (where the $i^{th}$ element of $\vec{g}$ is $g_i(v_i)$ ) are all the same sigmoid function. We will examine two ways of defining functions $\vec{f}$ given a function $F$. Along with these definitions will be

defined corresponding functions $E$ that will be used to describe the dynamics of Eq. (1).

The first method corresponds to optimization methods introduced by artificial neural network research. It will be referred to as method $\nabla_{\vec{g}}$ ("dell g"):

$$\vec{f} \equiv \nabla_{\vec{g}} F \tag{2a}$$

with associated $E$ function

$$E_{\vec{g}} = F(\vec{g}) - \int^t \sum_i^N \left[ D^{(M)}(v_i(s)) - \frac{dv_i(s)}{dt} \right] \frac{dg_i(s)}{dt} \, ds. \tag{2b}$$

Here, $\nabla_{\vec{x}} H$ denotes the gradient of $H$, where partials are taken with respect to variables of $\vec{x}$, and $E_{\vec{x}}$ denotes the $E$ function associated with gradient operator $\nabla_{\vec{x}}$. With appropriate operator $D$ and functions $\vec{f}$ and $\vec{g}$, $E_{\vec{g}}$ is simply the "energy function" of Hopfield [12]. Note that Eq. (2a) makes explicit that we will only be concerned with $\vec{f}$ that can be derived from scalar potential functions. For example, this restriction excludes artificial neural networks that have connections between excitatory and inhibitory units such as that of Freeman [8]. The second method corresponds to optimization methods based on the gradient. It will be referred to as method $\nabla_{\vec{v}}$ ("dell v"):

$$\vec{f} \equiv \nabla_{\vec{v}} F \tag{3a}$$

with associated $E$ function

$$E_{\vec{v}} = F(\vec{g}) - \int^t \sum_i^N \left[ D^{(M)}(v_i(s)) - \frac{dv_i(s)}{dt} \right] \frac{dv_i(s)}{dt} \, ds \tag{3b}$$

where notation is analogous to that for Eqs. (2).

The critical result that allows us to map optimization problems into networks described by Eq. (1) is that conditions on the constituents of the equation can be chosen so that along any solution trajectory, the $E$ function corresponding to the system will be a monotonic function of time. For method $\nabla_{\vec{g}}$, here are the conditions: all functions $\vec{g}$ are 1) differentiable and 2) monotonic in the same sense. Only the first condition is needed to make a similar assertion for

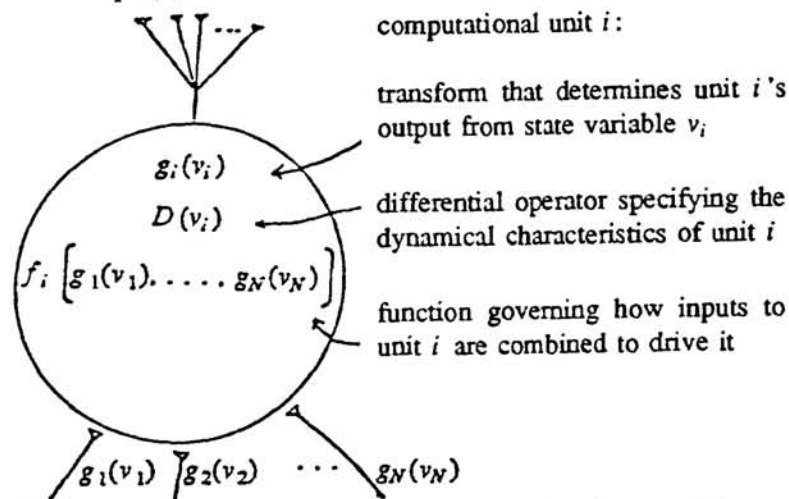

computational unit $i$:

transform that determines unit $i$'s output from state variable $v_i$

differential operator specifying the dynamical characteristics of unit $i$

function governing how inputs to unit $i$ are combined to drive it

Figure 1: Schematic of a computational unit $i$ from which networks considered in this paper are constructed. Triangles suggest connections between computational units.

method $\nabla_{\vec{v}}$. When these conditions are met and when solutions of Eq. (1) exist, the dynamical systems can be used for optimization. The appendix contains proofs for the monotonicity of function $E$ along solution trajectories and references necessary existence theorems. In conclusion, mapping optimization problems onto dynamical systems summarized by Eq. (1) can be reduced to a matter of differentiation if a scalar function representation of the problem can be found and the integrals of Eqs. (2b) and (3b) are ignorable. This last assumption is certainly upheld for the case where operator $D$ has no derivatives less than $M^{th}$ order. In simulations below, it will be observed to hold for the case $M=1$ with a nonzero $0^{th}$ order derivative in $D$. (Also see Lapedes and Farber [19].)

**PERSPECTIVES OF RECENT WORK**

The formulations above can be used to classify the neural network optimization techniques used in several recent studies. In these studies, the functions $\vec{g}$ were all identical. For the most part, following Hopfield's formulation, researchers [10, 13, 14, 17, 23, 24] have used method $\nabla_{\vec{g}}$ to derive forms of Eq. (1) that exhibit the ability to find extrema of $E_{\vec{g}}$ with $E_{\vec{g}}$ quadratic in functions $\vec{g}$ and all functions $\vec{g}$ describable by sigmoid functions such as $tanh(x)$. However, several researchers have written about artificial neural networks associated with non-quadratic $E$ functions. Method $\nabla_{\vec{g}}$ has been used to derive systems capable of finding extrema of non-quadratic $E_{\vec{g}}$ [19]. Method $\nabla_{\vec{v}}$ has been used to derive systems capable of optimizing $E_{\vec{v}}$ where $E_{\vec{v}}$ were not necessarily quadratic in variables $\vec{v}$ [21]. A sort of hybrid of the two methods was used by Jeffery and Rosner [18] to find extrema of functions that were not quadratic. The important distinction is that their functions $\vec{f}$ were derived from a given function $F$ using Eq. (3a) where, in addition, a sign definite diagonal matrix was introduced; the left side of Eq. (3a) was left multiplied by this matrix. A perspective on the relationship between all three methods to construct dynamical systems for optimization is summarized by Eq. (4) which describes the relationship between methods $\nabla_{\vec{g}}$ and $\nabla_{\vec{v}}$:

$$\nabla_{\vec{g}}F = diag\left[\frac{\partial g(v_i)}{\partial v_i}\right]^{-1} \nabla_{\vec{v}}F \tag{4}$$

where $diag[x_i]$ is a diagonal matrix with $x_i$ as the diagonal element of row $i$. (A similar equation has been derived for quadratic $F$ [5].) The relationship between the method of Jeffery and Rosner and $\nabla_{\vec{v}}$ is simply Eq. (4) with the time dependent diagonal matrix replaced by a constant diagonal matrix of free parameters. It is noted that Jeffery and Rosner presented timing results that compared simulated annealing, conjugate-gradient, and artificial neural network methods for optimization. Their results are not comparable to the results reported below since they used computation time as a performance measure, not settling times of analog systems. The perspective provided by Eq. (4) will be useful for anticipating the relative performance of methods $\nabla_{\vec{g}}$ and $\nabla_{\vec{v}}$ in the analytical example below and will aid in understanding the results of computer simulations.

## COMPARISON OF METHODS $\nabla_{\vec{g}}$ AND $\nabla_{\vec{v}}$

When $M=1$ and operator $D$ has no $0^{th}$ order derivatives, method $\nabla_{\vec{v}}$ is the basis of gradient-search methods of optimization. Given the long history of of such methods, it is important to know what possible benefits could be achieved by the relatively new optimization scheme, method $\nabla_{\vec{g}}$. In the following, the optimization efficiency of methods $\nabla_{\vec{g}}$ and $\nabla_{\vec{v}}$ is compared by comparing settling times, the time required for dynamical systems described by Eq. (1) to traverse a continuous path to local optima. To qualify this performance measure, this study anticipates application to the creation of analog devices that would instantiate Eq. (1); hence, we are not interested in estimating the number of discrete steps that would be required to find local optima, an appropriate performance measure if the point was to develop new numerical methods. An analytical example will serve to illustrate the possibility of improvements in settling time by using method $\nabla_{\vec{g}}$ instead of method $\nabla_{\vec{v}}$. Computer simulations will be reported for more complicated problems following this example.

For the analytical example, we will examine the case where all functions $\vec{g}$ are identical and

$$g(v) = tanhG(v-Th) \tag{5}$$

where $G > 0$ is the gain and $Th$ is the threshold. Transforms similar to this are widely used in artificial neural network research. Suppose we wish to use such computational units to search a multi-dimensional binary solution space. We note that

$$\frac{dg}{dv} = G\,sech^2G(v-Th) \tag{6}$$

is near 0 at valid solution states (corners of a hypercube for the case of binary solution spaces). We see from Eq. (4) that near a valid solution state, a network based on method $\nabla_{\vec{g}}$ will allow computational units to recede from incorrect states and approach correct states comparatively faster. Does

this imply faster settling time for method $\nabla_{\vec{z}}$?

To obtain an analytical comparison of settling times, consider the case where $M=1$ and operator $D$ has no $0^{th}$ order derivatives and

$$F = \frac{1}{2}\sum_{i,j}S_{ij}(tanhGv_i)(tanhGv_j) \tag{7}$$

where matrix S is symmetric. Method $\nabla_{\vec{z}}$ gives network equations

$$\frac{d\vec{v}}{dt} = S\,tanhG\vec{v} \tag{8}$$

and method $\nabla_{\vec{v}}$ gives network equations

$$\frac{d\vec{v}}{dt} = diag\,[G\,sech^2Gv_i]\,S\,tanhG\vec{v} \tag{9}$$

where $tanhG\vec{v}$ denotes a vector with $i^{th}$ component $tanhGv_i$. For method $\nabla_{\vec{z}}$, there is one stable point, i.e. where $\frac{d\vec{v}}{dt} = 0$, at $\vec{v} = 0$. For method $\nabla_{\vec{v}}$ the stable points are $\vec{v} = 0$ and $\vec{v} \in V$ where $V$ is the set of vectors with component values that are either $+\infty$ or $-\infty$. Further trivialization allows for comparing estimates of settling times: Suppose S is diagonal. For this case, if $v_i = 0$ is on the trajectory of any computational unit $i$ for one method, $v_i = 0$ is on the trajectory of that unit for the other method; hence, a comparison of settling times can be obtained by comparing time estimates for a computational unit to evolve from near 0 to near an extremum or, equivalently, the converse. Specifically, let the interval be $[\delta_0, 1-\delta]$ where $0<\delta_0<1-\delta$ and $0<\delta<1$. For method $\nabla_{\vec{v}}$, integrating velocity over time gives the estimate

$$T_{\nabla_{\vec{v}}} = \frac{1}{G}\left[\frac{1}{2}\left[\frac{1}{\delta(2-\delta)} - \frac{1}{1-\delta_0^2}\right] + \ln\left[\frac{1-\delta}{\sqrt{\delta(2-\delta)}}\frac{\sqrt{1-\delta_0^2}}{\delta_0}\right]\right] \tag{10}$$

and for method $\nabla_{\vec{z}}$ the estimate is

$$T_{\nabla_{\vec{z}}} = \frac{1}{G}\ln\left[\frac{1-\delta}{\sqrt{\delta(2-\delta)}}\frac{\sqrt{1-\delta_0^2}}{\delta_0}\right] \tag{11}$$

From these estimates, method $\nabla_{\vec{v}}$ will always take longer to satisfy the criterion for convergence: Note that only with the largest value for $\delta_0$, $\delta_0 = 1-\delta$, is the first term of Eq. (10) zero; for any smaller $\delta_0$, this term is positive. Unfortunately, this simple analysis cannot be generalized to non-diagonal S. With diagonal S, all computational units operate independently. Hence, the derivation of $\frac{d\vec{v}}{dt}$ is irrelevant with respect to convergence rates; convergence rate depends only on the diagonal element of S having the smallest magnitude. In this sense, the problem is one dimensional. But for non-diagonal S, the problem would be, in general, multi-dimensional and, hence, the direction of $\frac{d\vec{v}}{dt}$ becomes relevant. To compare settling times for non-diagonal S, computer simulations were done. These are described below.

## COMPUTER SIMULATIONS

*Methods*

The problem chosen for study was a much simplified version of a problem in medical imaging: Given electromagnetic field measurements taken from the human scalp, identify the location and magnitude of cerebral activity giving rise to the fields. This problem has received much attention in the last 20 years [3,6,7]. The problem, sufficient for our purposes here, was reduced to the following problem: given a few samples of the electric potential field at the surface of a spherical conductor within which reside several static electric dipoles, identify the dipole locations and moments. For this situation, there is a closed form solution for electric potential fields at the spherical surface:

$$\Phi(\vec{x}_{sample}) = \sum_{\substack{all\ dipoles \\ i}} \vec{p}_i \bullet \left[ \frac{2\hat{d}_i}{d_i} + \frac{\hat{x}_{sample} + \hat{d}_i}{1 + \hat{x}_{sample} \bullet \hat{d}_i} \frac{1}{x_{sample}} \right] \frac{1}{d_i} \qquad (12)$$

where $\Phi$ is the electric potential at the spherical conductor surface, $\vec{x}_{sample}$ is the location of the sample point ( $\vec{x}$ denotes a vector, $\hat{x}$ the corresponding unit vector, and $x$ the corresponding vector magnitude), $\vec{p}_i$ is the dipole moment of dipole $i$, and $\vec{d}_i$ is the vector from dipole $i$ to $\vec{x}_{sample}$ (This equation can be derived from one derived by Brody, Terry, and Ideker [4] ). Fig. 2 facilitates picturing these relationships.

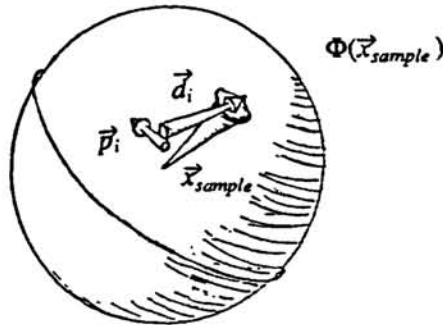

Figure 2: Vectors of Eq. (12).

With this analytical solution, the problem was formulated as a least squares minimization problem where the variables were dipole moments. In short, the following process was used: A dipole model was chosen. This model was used with Eq. (12) to calculate potentials at points on a sphere which covered about 60% of the surface. A cluster of internal locations that encompassed the locations of the model was specified. The two optimization techniques were then required to determine dipole moment values at cluster locations such that the collection of dipoles at cluster locations accurately reflected the dipole distribution specified by the model.

This was to be done given only the potential values at the sample points and an initial guess of dipole moments at cluster locations. The optimization systems were to accomplish the task by minimizing the sum of squared differences between potentials calculated using the dipole model and potentials calculated using a guess of dipole moments at cluster locations where the sum is taken over all sample points. Further simplifications of the problem included
1) choosing the dipole model locations to correspond exactly to various locations of the cluster,
2) requiring dipole model moments to be 1, 0, or -1, and
3) representing dipole moments at cluster locations with two bit binary numbers.

To describe the dynamical systems used, it suffices to specify operator $D$ and functions $\vec{g}$ of Eq. (1) and function $F$ used in Eqs. (2a) and (3a). Operator $D$ was

$$D = \frac{d}{dt} + 1. \qquad (13)$$

Eq. (5) with a multiplicative factor of ½ was used for all functions $\vec{g}$. Hence, regarding simplification 3) above, each cluster location was associated with two computational units. Considering simplification 2) above, dipole moment magnitude 1 would be represented by both computational units being in the high state, for -1, both in the low state, and for 0, one in the high state and one in the low state. Regarding function $F$,

$$F = \sum_{\substack{all\ sample \\ points\ s}} \left[ \Phi_{measured}(\vec{x}_s) - \Phi_{cluster}(\vec{x}_s) \right]^2 - c \sum_{\substack{all\ computational \\ units\ j}} g(v_j)^2 \qquad (14)$$

where $\Phi_{measured}$ is calculated from the dipole model and Eq. (12) (The subscript $measured$ is used because the role of the dipole model is to simulate electric potentials that would be measured in a real world situation. In real world situations, we do not know the source distribution underlying $\Phi_{measured}$.), $c$ is an experimentally determined constant (.002 was used), and $\Phi_{cluster}$ is Eq. (12) where the sum of Eq. (12) is taken over all cluster locations and the $k^{th}$ coordinate of the $i^{th}$ cluster location dipole moment is

$$p_{ik} = \sum_{all\ bits\ b} g(v_{ikb}). \qquad (15)$$

Index $j$ of Eq. (14) corresponds to one combination of indices *ikb*.

Sample points, 100 of them, were scattered semi-uniformly over the spherical surface emphasized by horizontal shading in Fig. 3. Cluster locations, 11, and model dipoles, 5, were scattered within the subset of the sphere emphasized by vertical shading. For the dipole model used, 10 dipole moment components were non-zero; hence, optimization techniques needed to hold 56 dipole moment components at zero and set 10 components to correct non-zero values in order to correctly identify the dipole model underlying $\Phi_{measured}$.

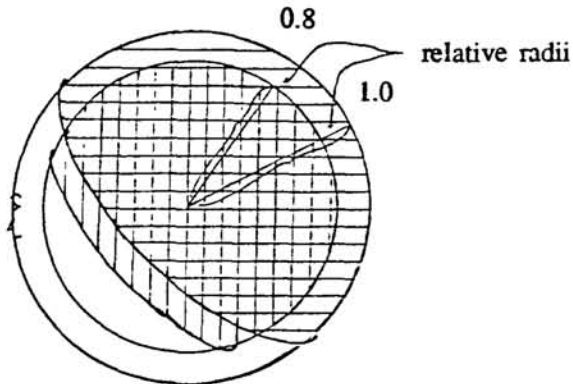

0.8
relative radii
1.0

Figure 3: Illustration of the distribution of sample points on the surface of the spherical conductor (horizontal shading) and the distribution of model dipole locations and cluster locations within the conductor (vertical shading).

The dynamical systems corresponding to methods $\nabla_{\vec{g}}$ and $\nabla_{\vec{v}}$ were integrated using the forward Euler method (e.g. Press, Flannery, Teukolsky, and Vetterling [22]). Numerical methods were observed to be convergent experimentally: settling time and path length were observed to asymtotically approach stable values as step size of the numerical integrator was decreased over two orders of magnitude.

Settling times, path lengths, and relative directions of travel were calculated for the two optimization methods using several different initial bit patterns at the cluster locations. In other words, the search was started at different corners of the hypercube comprising the space of acceptable solutions. One corner of the hypercube was chosen to be the target solution. (Note that a zero dipole moment has a degenerate two bit representation in the dynamical systems explored; the target corner was arbitrarily chosen to be one of the degenerate solutions.) Note from Eq. (5) that for the network to reach a hypercube corner, all elements of $\vec{v}$ would have to be singular. For this reason, settling time and other measures were studied as a function of the proximity of the computational units to their extremum states.

Computations were done on a Sequent Balance.

*Results*

Graph 1 shows results for exploring settling time as a function of *extremum depth*, the minimum of the deviations of variables $\vec{v}$ from the threshold of functions $\vec{g}$. Extremum depth is reported in multiples of the width of functions $\vec{g}$. The term *transition*, used in the caption of Graph 1 and below, refers to the movement of a computational unit from one extremum state to the other. The calculations were done for two initial states, one where the output of 1 computational unit was set to zero and one where outputs of 13 computational units were set to zero; hence, 1 and 13, respectively, half transitions were required to reach the target hypercube corner. It can be observed that settling time increases faster for method $\nabla_{\vec{v}}$ than that for method $\nabla_{\vec{g}}$ just as we would expect from considering Eqs. (4) and (5). However, it can be observed that method $\nabla_{\vec{v}}$ is still an order of magnitude faster even when extremum depth is 3 widths of functions $\vec{g}$. For the purpose of unambiguously identifying what hypercube corner the dynamical system settles

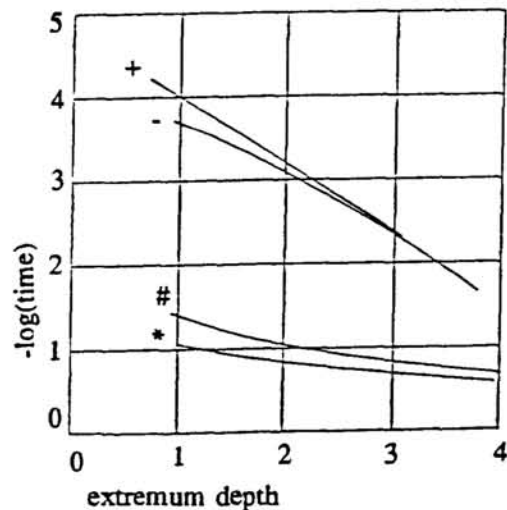

-log(time)

extremum depth

Graph 1: settling time as a function of extremum depth. #: method $\nabla_{\vec{v}}$, 1 half transition required. *: method $\nabla_{\vec{v}}$, 13 half transitions required. +: method $\nabla_{\vec{g}}$, 1 half transition required. -: $\nabla_{\vec{g}}$, 13 half transitions required.

to, this extremum depth is more than adequate.

Table 1 displays results for various initial conditions. Angles are reported in degrees. These measures refer to the angle between directions of travel in $\vec{v}$-space as specified by the two optimization methods. The average angle reported is taken over all trajectory points visited by the numerical integrator. Initial angle is the angle at the beginning of the path. *Parasite cost percentage* is a measure that compares parasite cost, the integral in Eqs. (2b) and (3b), to the range of function $F$ over the path:

$$parasite\ cost\ \% = 100 \times \frac{parasite\ cost}{|F_{final} - F_{initial}|} \qquad (16)$$

| transitions required | time | relative time | path length | initial angle | Mean angle (std dev) | extremum depth | parasite cost % |
|---|---|---|---|---|---|---|---|
| 1 | 0.16 0.0016 | 100 | 6.1 1.9 | 68 | 76 (3.8) 76 (3.5) | 2.3 2.3 | 0.22 0.039 |
| 2 | 0.14 0.0018 | 78 | 4.7 1.9 | 75 | 72 (4.3) 73 (4.1) | 2.5 2.5 | 0.055 0.016 |
| 3 | 0.15 0.0021 | 71 | 4.7 2.1 | 74 | 71 (3.7) 72 (3.0) | 2.3 2.5 | 0.051 0.0093 |
| 7 | 0.19 0.0032 | 59 | 4.6 2.4 | 63 | 69 (4.1) 71 (7.0) | 2.4 2.7 | 0.058 0.0033 |
| 10 | 0.17 0.0035 | 49 | 3.8 2.5 | 60 | 63 (2.8) 64 (4.7) | 2.5 2.8 | 0.030 0.00060 |
| 13 | 0.80 0.0074 | 110 | 9.2 3.2 | 39 | 77 (11) 71 (8.9) | 2.3 2.7 | 0.076 0.0028 |

Table 1: Settling time and other measurements for various required transitions. For each transition case, the upper row is for $\nabla_{\vec{z}}$ and the lower row is for $\nabla_{\vec{v}}$. *Std dev* denotes standard deviation. See text for definition of measurement terms and units.

Noting the differences in path length and angles reported, it is clear that the path taken to the target hypercube corner was quite different for the two methods. Method $\nabla_{\vec{v}}$ settles from 1 to 2 orders of magnitude faster than method $\nabla_{\vec{z}}$ and usually takes a path less than half as long. These relationships did not change significantly for different values for $c$ of Eq. (14) and coefficients of Eq. (13) (both unity in Eq. (13)). Values used favored method $\nabla_{\vec{z}}$. Parasite cost is consistently less significant for method $\nabla_{\vec{v}}$ and is quite small for both methods.

To further compare the ability of the optimization methods to solve the brain imaging problem, a large variety of initial hypercube corners were tested. Table 2 displays results that suggest the ability of each method to locate the target corner or to converge to a solution that was consistent with the dipole model. Initial corners were chosen by randomly selecting a number of computational units and setting them to extremum states opposite to that required by the target solution. Five cases were run for each case of required transitions. It can be observed that the system based on method $\nabla_{\vec{v}}$ is better at finding the target corner and is much better at finding a solution that is consistent with the dipole model.

## DISCUSSION

The simulation results seem to contradict settling time predictions of the second analytical example. It is intuitively clear that there is no contradiction when considering the analytical example as a one dimensional search and the simulations as multi-dimensional searches. Consider Fig. 4 which illustrates one dimensional search starting at point $I$. Since both optimization methods must decrease function $E$ monotonically, both must head along the same path to the minimum point $A$. Now consider Fig. 5 which illustrates a two dimensional search starting at point $I$: Here, the two methods needn't follow the same paths. The two dashed paths suggest that method $\nabla_{\vec{z}}$ can still be

| transitions required | $\nabla_{\vec{z}}$ | | | $\nabla_{\vec{v}}$ | | |
|---|---|---|---|---|---|---|
| | different dipole solution | different corner | target corner | different dipole solution | different corner | target corner |
| 3 | 1 | 0 | 4 | 0 | 0 | 5 |
| 4 | 1 | 1 | 3 | 0 | 1 | 4 |
| 5 | 0 | 1 | 4 | 0 | 1 | 4 |
| 6 | 2 | 1 | 2 | 0 | 1 | 4 |
| 7 | 4 | 0 | 1 | 0 | 1 | 4 |
| 13 | 5 | 0 | 0 | 1 | 3 | 1 |
| 20 | 5 | 0 | 0 | 0 | 5 | 0 |
| 26 | 5 | 0 | 0 | 2 | 3 | 0 |
| 33 | 5 | 0 | 0 | 3 | 2 | 0 |
| 40 | 5 | 0 | 0 | 3 | 2 | 0 |
| 46 | 5 | 0 | 0 | 2 | 3 | 0 |
| 53 | 5 | 0 | 0 | 4 | 1 | 0 |

Table 2: Solutions found starting from various initial conditions, five cases for each transition case. *Different dipole solution* indicates that the system assigned non-zero dipole moments at cluster locations that did not correspond to locations of the dipole model sources. *Different corner* indicates the solution was consistent with the dipole model but was not the target hypercube corner. *Target corner* indicates that the solution was the target solution.

monotonically decreasing $E$ while traversing a more circuitous route to minimum $B$ or traversing a path to minimum $A$. The longer path lengths reported in Table 1 for method $\nabla_{\vec{v}}$ suggest the occurrence of the former. The data of Table 2 verifies the occurrence of the latter: Note that for many cases where the system based on method $\nabla_{\vec{v}}$ settled to the target corner, the system based on method $\nabla_{\vec{z}}$ settled to some other minimum.

Would we observe similar differences in optimization efficiency for other optimization problems that also have binary solution spaces? A view that supports the plausibility of the affirmative is the following: Consider Eq. (4) and Eq. (5). We have already made the observation that method $\nabla_{\vec{v}}$ would slow convergence into extrema of functions $\vec{g}$. We have observed this experimentally via Graph 1. These observations suggest that computational units of $\nabla_{\vec{v}}$ systems tend to stay closer to the transition regions of functions $\vec{g}$ compared to computational units of $\nabla_{\vec{z}}$ systems. It seems plausible that this property may allow $\nabla_{\vec{v}}$ systems to avoid advancing too deeply toward ineffective solutions and, hence, allow the systems to approach effective solutions more efficiently. This behavior might also be the explanation for the comparative success of method $\nabla_{\vec{v}}$ revealed in Table 2.

Regarding the construction of electronic circuitry to instantiate Eq. (1), systems based on method $\nabla_{\vec{v}}$ would require the introduction of a component implementing multiplication by the derivative of functions $\vec{g}$. This additional complexity may hinder the use of method $\nabla_{\vec{v}}$ for the

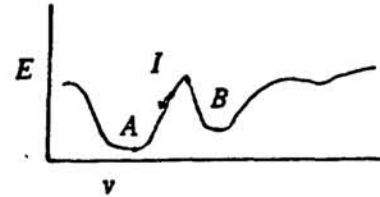

Figure 4: One dimensional search for minima.

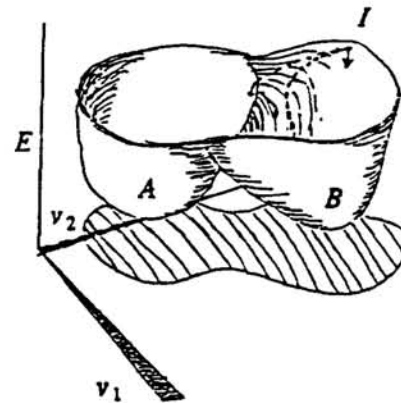

Figure 5: Two dimensional search for minima.

(a)
(b)

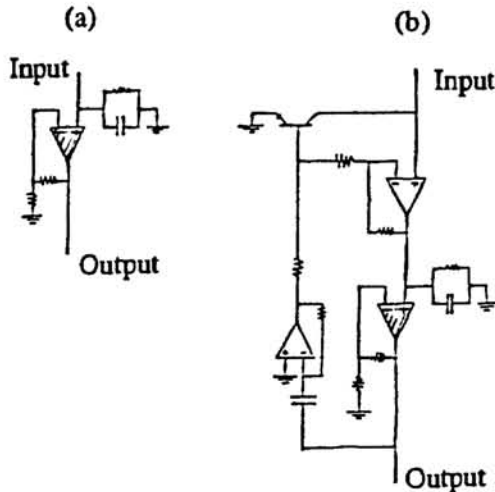

Input

Output

Input

Output

Figure 6: Schematized circuits for a computational unit. Notation is consistent with Horowitz and Hill [15]. Shading of amplifiers is to earmark components referred to in the text. a) Computational unit for method $\nabla_{\vec{g}}$. b) Computational unit for method $\nabla_{\vec{v}}$.

construction of analog circuits for optimization. To illustrate the extent of this additional complexity, Fig. 6a shows a schematized circuit for a computational unit of method $\nabla_{\vec{g}}$ and Fig. 6b shows a schematized circuit for a computational unit of method $\nabla_{\vec{v}}$. The simulations reported above suggest that there may be problems for which improvements in settling time may offset complications that might come with added circuit complexity.

On the problem of imaging cerebral activity, the results above suggest the possibility of constructing analog devices to do the job. Consider the problem of analyzing electric potentials from the scalp of one person: It is noted that the measured electric potentials, $\Phi_{measured}$, appear as linear coefficients in $F$ of Eq. (14); hence, they would appear as constant terms in $\vec{f}$ of Eq. (1). Thus, $\Phi_{measured}$ would be implemented as amplifier biases in the circuits of Figs. 6. This is a significant benefit. To understand this, note that function $f_i$ of Fig. 1 corresponding to the optimization of function $F$ of Eq. (14) would involve a weighted linear sum of inputs $g_1(v_1),...,g_N(v_N)$. The weights would be the nonlinear coefficients of Eq. (14) and correspond to the strengths of the connections shown in Fig. 1. These connection strengths need only be calculated once for the person and can then be set in hardware using, for example, a resistor network. Electric potential measurements could then be analyzed by simply using the measurements to bias the input to shaded amplifiers of Figs. 6. For initialization, the system can be initialized with all dipole moments at zero (the 10 transition case in Table 1). This is a reasonable first guess if it is assumed that cluster locations are far denser than the loci of cerebral activity to be observed. For subsequent measurements, the solution for immediately preceding measurements would be a reasonable initial state if it is assumed that cerebral activity of interest waxes and wanes continuously.

Might non-invasive real time imaging of cerebral activity be possible using such optimization devices? Results of this study are far from adequate for answering this question. Many complexities that have been avoided may nullify the practicality of the idea. Among these problems are:

1) The experiment avoided the possibility of dipole sources actually occurring at locations other than cluster locations. The minimization of function $F$ of Eq. (14) may circumvent this problem by employing the superposition of dipole moments at neighboring cluster locations to give a sufficient model in the mean.

2) The experiment assumed a very restricted range of dipole strengths. This might be dealt with by increasing the number of bits used to represent dipole moments.

3) The conductor model, a homogeneously conducting sphere, may not be sufficient to model the human head [16]. Non-sphericity and major inhomogeneities in conductivity can be dealt with, to a certain extent, by replacing Eq. (12) with a generalized equation based on a numerical approximation of a boundary integral equation [20]

4) The cerebral activity of interest may not be observable at the scalp.

5) Not all forms of cerebral activity give rise to dipolar sources. (For example, this is well known in olfactory cortex [8].)

6) Activity of interest may be overwhelmed by irrelevant activity. Many methods have been devised to contend with this problem (For example, Gevins and Morgan [9].)

Clearly, much theoretical work is left to be done.

**CONCLUDING REMARKS**

In this study, the mapping principle underlying the application of artificial neural networks to the optimization of multi-dimensional scalar functions has been stated explicitly. Hopfield [12] has shown that for some scalar functions, i.e. functions $F$ quadratic in functions $\vec{g}$, this mapping can lead to dynamical systems that can be easily implemented in hardware, notably, hardware that requires electronic components common to semiconductor technology. Here, mapping principles that have been known for a considerably longer period of time, those underlying gradient based optimization, have been shown capable of leading to dynamical systems that can also be implemented using semiconductor hardware. A problem in medical imaging which requires the search of a multi-dimensional surface full of local extrema has suggested the superiority of the latter mapping principle with respect to settling time of the corresponding dynamical system. This advantage may be quite significant when searching for global extrema using techniques such as iterated descent [2] or iterated genetic hill climbing [1] where many searches for local extrema are required. This advantage is further emphasized by the brain imaging problem: volumes of measurements can be analyzed without reconfiguring the interconnections between computational units; hence, the cost of developing problem specific hardware for finding local extrema may be justifiable. Finally, simulations have contributed plausibility to a possible scheme for non-invasively imaging cerebral activity.

## APPENDIX

To show that for a dynamical system based on method $\nabla_{\vec{g}}$, $E_{\vec{g}}$ is a monotonic function of time given that all functions $\vec{g}$ are differentiable and monotonic in the same sense, we need to show that the derivative of $E_{\vec{g}}$ with respect to time is semi-definite:

$$\frac{dE_{\vec{g}}}{dt} = \sum_i^N \frac{\partial F_{\vec{g}}}{\partial g_i} \frac{dg_i}{dt} - \sum_i^N \left[ D^{(M)}(v_i) - \frac{dv_i}{dt} \right] \frac{dg_i}{dt}. \tag{A1a}$$

Substituting Eq. (2a),

$$\frac{dE_{\vec{g}}}{dt} = \sum_i^N \left[ f_i - D^{(M)}(v_i) + \frac{dv_i}{dt} \right] \frac{dg_i}{dt}. \tag{A1b}$$

Using Eq. (1),

$$\frac{dE_{\vec{g}}}{dt} = \sum_i^N \left[ \frac{dv_i}{dt} \right]^2 \frac{\partial g_i}{\partial v_i} \begin{smallmatrix} \geq \\ \leq \end{smallmatrix} 0 \tag{A1c}$$

as needed. The appropriate inequality depends on the sense in which functions $\vec{g}$ are monotonic. In a similar manner, the result can be obtained for method $\nabla_{\vec{v}}$. With the condition that functions $\vec{g}$ are differentiable, we can show that the derivative of $E_{\vec{v}}$ is semi-definite:

$$\frac{dE_{\vec{v}}}{dt} = \sum_i^N \frac{\partial F_{\vec{v}}}{\partial v_i} \frac{dv_i}{dt} - \sum_i^N \left[ D^{(M)}(v_i) - \frac{dv_i}{dt} \right] \frac{dv_i}{dt}. \tag{A2a}$$

Using Eqs. (3a) and (1),

$$\frac{dE_{\vec{v}}}{dt} = \sum_i^N \left[ \frac{dv_i}{dt} \right]^2 \begin{smallmatrix} \geq \\ \leq \end{smallmatrix} 0 \tag{A2b}$$

as needed.

In order to use the results derived above to conclude that Eq. (1) can be used for optimization of functions $E_{\vec{v}}$ and $E_{\vec{g}}$ in the vicinity of some point $\vec{v}_0$, we need to show that there exists a neighborhood of $\vec{v}_0$ in which there exist solution trajectories to Eq. (1). The necessary existence theorems and transformations of Eq. (1) needed in order to apply the theorems can be found in many texts on ordinary differential equations; e.g. Guckenheimer and Holmes [11]. Here, it is mainly important to state that the theorems require that functions $\vec{f} \in C^{(1)}$, functions $\vec{g}$ are differentiable, and initial conditions are specified for all derivatives of lower order than $M$.

ACKNOWLEDGEMENTS

I would like to thank Dr. Michael Raugh and Dr. Pentti Kanerva for constructive criticism and support. I would like to thank Bill Baird and Dr. James Keeler for reviewing this work. I would like to thank Dr. Derek Fender, Dr. John Hopfield, and Dr. Stanley Klein for giving me opportunities that fostered this conglomeration of ideas.

## Footnotes

† Work supported by NASA Cooperative Agreement No. NCC 2-408

## REFERENCES

[1]   Ackley D.H., "Stochastic iterated genetic hill climbing", PhD. dissertation, Carnegie Mellon U., 1987.

[2]   Baum E., Neural Networks for Computing, ed. Denker J.S. (AIP Confrnc. Proc. 151, ed. Lerner R.G.), p53-58, 1986.

[3]   Brody D.A., IEEE Trans. vBME-32, n2, p106-110, 1968.

[4]   Brody D.A., Terry F.H., Ideker R.E., IEEE Trans. vBME-20, p141-143, 1973.

[5]   Cohen M.A., Grossberg S., IEEE Trans. vSMC-13, p815-826, 1983.

[6]   Cuffin B.N., IEEE Trans. vBME-33, n9, p854-861, 1986.

[7]   Darcey T.M., Ary J.P., Fender D.H., Prog. Brain Res., v54, p128-134, 1980.

[8]   Freeman W.J., "Mass Action in the Nervous System", Academic Press, Inc., 1975.

[9]   Gevins A.S., Morgan N.H., IEEE Trans., vBME-33, n12, p1054-1068, 1986.

[10]  Goles E., Vichniac G.Y., Neural Networks for Computing, ed. Denker J.S. (AIP Confrnc. Proc. 151, ed. Lerner R.G.), p165-181, 1986.

[11]  Guckenheimer J., Holmes P., "Nonlinear Oscillations, Dynamical Systems, and Bifurcations of Vector Fields", Springer Verlag, 1983.

[12]  Hopfield J.J., Proc. Natl. Acad. Sci., v81, p3088-3092, 1984.

[13]  Hopfield J.J., Tank D.W., Bio. Cybrn., v52, p141-152, 1985.

[14]  Hopfield J.J., Tank D.W., Science, v233, n4764, p625-633, 1986.

[15]  Horowitz P., Hill W., "The art of electronics", Cambridge U. Press, 1983.

[16]  Hosek R.S., Sances A., Jodat R.W., Larson S.J., IEEE Trans., vBME-25, n5, p405-413, 1978.

[17]  Hutchinson J.M., Koch C., Neural Networks for Computing, ed. Denker J.S. (AIP Confrnc. Proc. 151, ed. Lerner R.G.), p235-240, 1986.

[18]  Jeffery W., Rosner R., Astrophys. J., v310, p473-481, 1986.

[19]  Lapedes A., Farber R., Neural Networks for Computing, ed. Denker J.S. (AIP Confrnc. Proc. 151, ed. Lerner R.G.), p283-298, 1986.

[20]  Leong H.M.F., "Frequency dependence of electromagnetic fields: models appropriate for the brain", PhD. dissertation, California Institute of Technology, 1986.

[21]  Platt J.C., Hopfield J.J., Neural Networks for Computing, ed. Denker J.S. (AIP Confrnc. Proc. 151, ed. Lerner R.G.), p364-369, 1986.

[22]  Press W.H., Flannery B.P., Teukolsky S.A., Vetterling W.T., "Numerical Recipes", Cambridge U. Press, 1986.

[23]  Takeda M., Goodman J.W., Applied Optics, v25, n18, p3033-3046, 1986.

[24]  Tank D.W., Hopfield J.J., "Neural computation by concentrating information in time", preprint, 1987.
